# Oriented Non-Radial Basis Functions for Image Coding and Analysis

**Avijit Saha**[1]     **Jim Christian**     **D. S. Tang**

*Microelectronics and Computer Technology Corporation*
*3500 West Balcones Center Drive*
*Austin, TX 78759*

**Chuan-Lin Wu**

*Department of Electrical and Computer Engineering*
*University of Texas at Austin,*
*Austin, TX 78712*

## ABSTRACT

We introduce oriented non-radial basis function networks (ONRBF) as a generalization of Radial Basis Function networks (RBF)- wherein the Euclidean distance metric in the exponent of the Gaussian is replaced by a more general polynomial. This permits the definition of more general regions and in particular- hyper-ellipses with orientations. In the case of hyper-surface estimation this scheme requires a smaller number of hidden units and alleviates the "curse of dimensionality" associated kernel type approximators.In the case of an image, the hidden units correspond to features in the image and the parameters associated with each unit correspond to the rotation, scaling and translation properties of that particular "feature". In the context of the ONBF scheme, this means that an image can be represented by a small number of features. Since, transformation of an image by rotation, scaling and translation correspond to identical transformations of the individual features, the ONBF scheme can be used to considerable advantage for the purposes of image recognition and analysis.

## 1   INTRODUCTION

Most, "neural network" or "connectionist" models have evolved primarily as adaptive function approximators. Given a set of input-output pairs <x,y> (x from an underlying function f (i.e. y = f(x)), a feed forward, time-independent neural network estimates a

---

1. Alternate address: Dept. of ECE, Univ. of Texas at Austin, Austin, TX 78712

function y' = g(**p**,**x**) such that E= ρ(y - y') is arbitrarily small over all <x,y> pairs. Here, **p** is the set of parameters associated with the network model and ρ is a metric that measures the quality of approximation, usually the Euclidean norm. In this paper, we shall restrict our discussion to approximation of real valued functions of the form f:$R^n$ -> R. For a network of fixed structure (determined by g), all or part of the constituent parameter set **p**, that minimize E are determined adaptively by modifying the set of parameters. The problem of approximation or hypersurface reconstruction is then one of determining what class of g to use, and then the choice of a suitable algorithm for determining the parameters **p** given a set of samples {<x,y>}.By far the most popular method for determining network parameters has been the gradient descent method. If the error surface is quadratic or convex, gradient descent methods will yield an optimal value for the network parameters.However, the burning problem in still remains the determination of network parameters when the error function is infested with local minimas. One way of obviating the problem of local minimas is to match a network architecture with an objective function such that the error surface is free of local minimas. However, this might limit the power of the network architecture such as in the case of linear perceptrons[1]. Another approach is to obtain algebraic transformations of the objective functions such that algorithms can be readily designed around the transformed functions to avoid local minimas. Random optimization method of Matyas and its variations have been studied recently [2], as alternate avenues for determining the parameter set **p**. Perhaps the most probable reason for the BP algorithms popularity is that the error surface is relatively smooth [1],[3]

The problem of local minimas is circumvented somewhat differently in local or kernel type estimators. The input space in such a method is partitioned into a number of local regions and if the number of regions defined is sufficiently large, then the output response in each local region is sufficiently uniform or smooth and the error will remain bounded i.e. a local minima will be close to the global minima. The problem with kernel type of estimators is that the number of "bins", "kernels" or "regions" that need to be defined increases exponentially with the dimension of the input space. An improvement such as the one considered by [4] is to define the kernels only in regions of the input space where there is data. However, our experiments indicate that even this may not be sufficient to lift the curse of dimensionality. If instead of limiting the shape of the kernels to be boxes or even hyperspheres we select the kernels to be shapes defined by a second order polynomials then a larger class of shapes or regions can be defined resulting in significant reductions in the number of kernels required. This was the principal motivation behind our generalization of ordinary RBF networks. Also, we have determined that radial basis function networks will, given sufficiently large widths, linearize the output response between two hidden units. This gives rise to hyperacuity or coarse coding, whereby a high resolution of stimuli can be observed at the signal level despite poor resolution in the sensor array. In the context of function approximation this means that if the hyper-surface being approximated varies linearly in a certain region, the output behavior can be captured by suitably placing a single widely tuned receptive field in that region. Therefore, it is advantageous to choose the regions with proper knowledge of the output response in that region as opposed to choosing the bins based on the inputs alone. These were some of the principal motivations for our generalization.

*In addition to the architectural and learning issues, we have been concerned with approximation schemes in which the optimal parameter values have readily interpretable forms that may allow other useful processing elsewhere.* In the following section we present ONBF as a generalization to RBF [4] and GRBF [5]. We show how rotation, scaling and

translation (center) information of these regions can be readily extracted from the parameter values associated with each hidden unit. In subsequent sections we present experimental results illustrating the performance of ONRBF as a function approximator and feasibility of ONRBF for the purposes of image coding and analysis.

## 2   ORIENTED NON-RADIAL BASIS FUNCTION NETWORKS

Radial Basis Function networks can be described by the formula:

$$f(x) = \sum_{\alpha=0}^{k} w_\alpha R_\alpha(x)$$

where $f(x)$ is the output of the network, $k$ is the number of hidden units, $w_\alpha$ is the weight associated with hidden unit $\alpha$, and $R_\alpha(x)$ is the response of unit $\alpha$, The response $R_\alpha(x)$ of unit $\alpha$ is given by

$$R_\alpha = e^{-\left(\frac{\|c_\alpha - x\|}{\sigma_\alpha}\right)^2}$$

Poggio and Girosi [5] have considered the generalization where a different width parameter $\sigma_{\alpha_i}$ is associated with each input dimension i. The response function $R_\alpha$ is then defined as

$$R_\alpha(x) = e^{-\sum_{i=1}^{d}\left(\frac{c_{\alpha_i} - x_i}{\sigma_{\alpha_i}}\right)^2}$$

Now each $\sigma_{\alpha_i}$ can influence the response of the $\alpha^{th}$ unit and the effect is that widths associated with irrelevant or correlated inputs will tend to be increased. It has been shown that if one of the input components has a random input and a constant width (constant for that particular dimension) is used for each receptive field, then the width for that particular receptive field is maximum [6].

The generalization we consider in this paper is a further shaping of the response $R_\alpha$ by composing it with a rotation function $S_\alpha$ designed to rotate the unit about its center in d-space, where d is the input dimension. This composition can be represented compactly by a response function of the form:

$$R_\alpha = e^{-\|M_\alpha[x_1,....x_d,1]\|^2}$$

where $M_\alpha$ is a d by d+1 matrix. The matrix transforms the input vectors and these transformations correspond to translation (center information), scaling and rotation of the input vectors. The response function presented above is the restricted form of a more general response function of the form:

$$R_\alpha = e^{-[P(x)]}$$

where the exponent is a general polynomial in the input variables. In the following sections we present the learning rules and we show how center, rotation and scaling information can be extracted from the matrix elements. We do this for the case when the input dimension is 2 (as is the case for 2-dimensional images) but the results are generalized easily.

## 2.1 LEARNING RULES

Consider the n-dimensional case where $<x_1,...x_n>$ represents the input vector and $m_{\alpha jk}$ represents the matrix element of the $j^{th}$ row and $k^{th}$ column of the matrix $M_\alpha$ associated with the $\alpha^{th}$ unit. Then the response of the $\alpha^{th}$ unit is given by:

$$R_\alpha(x,y) = e^{-\left(\sum_{j=1}^{n}\left(\sum_{i=1}^{n+1} m_{\alpha ji} x_i\right)\right)}$$

The total sum square error over b patterns is given by:

$$TE = \sum_\beta \left[f(x_\beta) - F(x_\beta)\right]^2 = \sum_\beta E_\beta = \sum_\beta L_\beta^2$$

Then the derivative of the error due to the $\beta^{th}$ pattern with respect to the matrix element $m_{\alpha ij}$ of the $\alpha^{th}$ unit is given by:

$$\therefore \frac{\partial}{\partial m_{\alpha_{ij}}}(E_\beta) = 2\,[f(x_\beta) - F(x_\beta)]\frac{\partial}{\partial m_{\alpha_{ij}}}f = 2\,[L_\beta]\frac{\partial}{\partial m_{\alpha_{ij}}}f$$

and:

$$\frac{\partial}{\partial m_{\alpha_{ij}}}f = -2m_{\alpha_i}\langle x_\beta, 1\rangle^T x_j R_\alpha(x_\beta)$$

where,

$m_{\alpha i}$ : is the $i^{th}$ row of the matrix corresponding to the $\alpha^{th}$ unit

$x_\beta$ : is the input vector

$x_j$ : is the $j^{th}$ variable in the input space.

Then the update rule for the matrix elements with learning rate $\eta$ is given by:

$$m_{\alpha_{ij}}^{t+1} = m_{\alpha_{ij}}^{t} - \eta\frac{\partial}{\partial m_{\kappa_{ij}}}(E_\beta)$$

and the learning rule for the weights $w_\alpha$ is given by:

$$w_\alpha^{t+1} = w_\alpha^t - \eta L_\beta R_\alpha(x_\beta)$$

## 2.2 EXTRACTING ROTATION, SCALE AND CENTER VALUES

In this section we present the equations for extracting the rotation, translation and scaling values (widths) of the $\alpha^{th}$ receptive field from its associated matrix elements. We present these for the special case when n the input dimension is equal to 2, since that is the case for images. The input vector x is represented by $<x,y>$ and the rules for converting the matrix elements into center, scaling and rotation information is as follows:

- center $(x_0, y_0)$

$$x_0 = \frac{(m_{12}m_{11} + m_{21}m_{22})(m_{11}m_{13} + m_{23}m_{21}) - (m_{11}^2 + m_{21}^2)(m_{13}m_{12} + m_{23}m_{22})}{\Delta}$$

$$y_0 = \frac{(m_{12}m_{11}+m_{21}m_{22})(m_{12}m_{13}+m_{23}m_{22})-(m_{12}^2+m_{22}^2)(m_{13}m_{11}+m_{23}m_{21})}{\Delta}$$

where,

$$\Delta = (m_{11}^2+m_{21}^2)(m_{12}^2+m_{22}^2)-(m_{11}m_{12}+m_{22}m_{21})$$

- rotation ($\theta$)

$$\theta = \frac{1}{2}\tan^{-1}\left(\frac{m_{21}m_{22}+m_{12}m_{11}}{m_{11}^2+(m_{21}^2-m_{12}^2-m_{22}^2)}\right)$$

- scaling or receptive field widths or sigmas

$$d_1 = \frac{1}{2}(m_{11}^2+m_{21}^2+m_{12}^2+m_{22}^2)+\frac{m_{12}m_{11}+m_{22}m_{21}}{\sin2\theta} \equiv \frac{1}{\sqrt{2}\sigma_1}$$

$$d_2 = \frac{1}{2}(m_{11}^2+m_{21}^2+m_{12}^2+m_{22}^2)-\frac{m_{12}m_{11}+m_{22}m_{21}}{\sin2\theta} \equiv \frac{1}{\sqrt{2}\sigma_2}$$

## 2.3 HIERARCHICAL CLUSTERING

We use a multi-resolution, hierarchical approach to determine where to place hidden units to maximize the accuracy of approximation and to locate image features. For illustration, we consider our method in the context of image processing, though the idea will work for any type of function approximation problem. The process begins with a small number of widely tuned receptive field units. The widths are made high my multiplying the value obtained from the nearest neighbor-heuristic by a large overlap parameter. The large widths force the units to excessively smooth the image being approximated. Then, errors will be observed in regions where detailed features occur. Those pixels for which high error (say, greater than one standard deviation from the mean) occurred are collected and new units are added in locations chosen randomly from this set. The entire process can be repeated until a desired level of accuracy is reached. Notice that, when the network is finally trained, the top levels in the hierarchy provide global information about the image under consideration. This scheme is slightly different than the one presented in [7], where units in each resolution learn the error observed in the previous resolution-- in our method, after the addition of the new units all the units learn the original function as opposed to the some error function.

## 3   RESULTS

### 3.1 ONRBF AS AN APPROXIMATOR

Oriented non-radial basis function networks allow the definition of larger regions or receptive fields- this is due to the fact that rotation, along with the elliptical hyper-spheres as opposed to mere spheres, permits the grouping of more nearby points into a single region.

Therefore, the approximation accuracy of such a network can be quite good with even a small number of units. For instance, Table 1 compares ordinary radial basis function networks with oriented non-radial basis function networks in terms of the number of units required to achieve various levels of accuracy. The function approximated is the Mackey-Glass differential delay equation:

$$\frac{dx_t}{dt} = -bx_t + a\frac{x_{t-\tau}}{1+x_{t-\tau}}$$

**TABLE 1. Normalized approximation error for radial and non-radial basis functions**

|  | RBF Train | ONBF Train | RBF Test 1 | ONBF Test 1 | RBF Test 2 | ONBF Test 2 |
|---|---|---|---|---|---|---|
| 10 units | .426 | .229 | .267 | .161 | .522 | .298 |
| 20 units | .377 | .119 | .167 | .071 | .497 | .166 |
| 40 units | .236 | .057 | .134 | .065 | .310 | .105 |
| 80 units | .197 |  | .123 |  | .271 |  |
| 160 units | .159 |  | .126 |  | .228 |  |
| 320 units | .107 |  | .131 |  | .207 |  |
| 500 units | .061 |  | .121 |  | .208 |  |

The series used was generated with $t = 17$, $a = 0.1$ and $b = 0.2$. A series of 500 consecutive points was used for training, and the next two sets of 500 points were used for cross-validation. The training vector at time t is the tuple $(x_t, x_{t-6}, x_{t-12}, x_{t-18}, x_{t+85})$, where the first four components form the input vector and the last forms the target, and $x_t$ is the value of the series at time t. Table 1 lists the normalized error for each experiment- that is, the root mean square prediction error divided by the standard deviation of the data series. Oriented non-radial basis function networks yield higher accuracy than do radial basis function networks with the same number of units. In addition, ONRBF nets were found to generalize better.

### 3.2 IMAGE CODING AND ANALYSIS

For images each hidden unit corresponds some feature in the input space. This implies that there is some invariant property associated with the region spanned by the receptive field. For bitmaps this property could be the probability density function (ignoring higher order statistics) and a feature is a region over which the probability density function remains the same. For grey level images, instead of the linear weight this property could be described by a low order polynomial. We have found that when the parameters of an image function are determined adaptively using the learning rules in section 2.1-- the receptive fields organize themselves so as to capture features in the input space. This is illustrated in Figure 1, where the input image is a bitmap for a set of Chinese characters. The property of a feature in this case is the value of the pixel (0 or 1) in the coordinate location specified by the input- and therefore a linear term (for the weight) as used in section 2.1 is sufficient. Figure 1.a is the input bitmap image and figure 1.b shows the plot of the regions of influence of the individual receptive fields. Notice that the individual receptive fields tend to become "responsible" for entire strokes of the character.

We would like to point out that if the initial positions of the hidden units are chosen randomly, then with each new start of the approximation process a single feature may be represented by a collection of hidden units in many different manners- and the task of

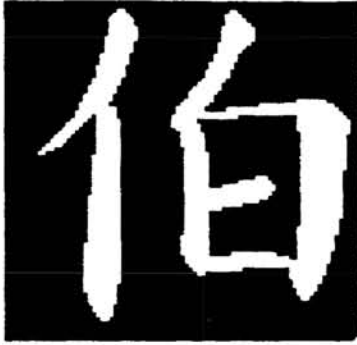

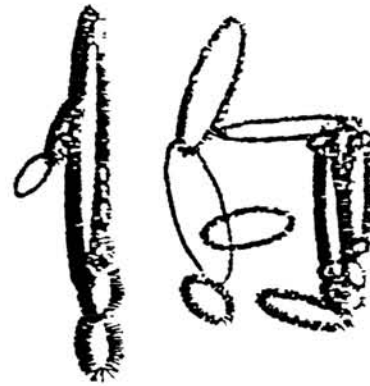

Figure 1.a: Bitmap Of
Chinese Character Which
Is The Input Image

Figure 1.b: Plot Of Regions Of Influence
Of Receptive Fields After Training

recognition becomes difficult. Therefore, for consistent approximation, a node deletion or region growing algorithm is needed. Such an algorithm has been developed and will be presented elsewhere. If with every approximation of the same image, we get the same features (parameters for the hidden units), then images under rotation and scaling can also be recognized easily-- since there will be a constant scaling and rotational change in all the hidden units.

## 4 CONCLUSIONS

We have presented a generalization of RBF networks that allows interpretation of the parameter values associated with the hidden units and performs better as a function approximator. The number of parameters associated with each hidden units grow quickly with the input dimension ($O(d^2)$). However, the number of hidden units required is significantly lower if the function is relatively smooth. Alternatively, one can compose the Gaussian response of the original RBF by using a suitable clipping function in which the number of associated parameters grow linearly with the input dimension d. For images, the input dimension is 2 and the number of parameters associated with each hidden unit is 6 as opposed to 5- when the multidimensional Gaussian is represented by the superposition of 1-dimensional Gaussians, and 4 with RBF networks.

### References

[1] Widrow, Bernard and Michael A. Lehr,"30 Years of Adaptive Neural Networks: Perceptron, Madaline, and Backpropagation", Proc. of the IEEE, vol.78, No. 9, Sept 1990, pp 1415-1442.

[2] Baba, Norio,"A New Approach for Finding the Global Minimum of Error Function of Neural Networks", Neural Networks, Vol. 2, pp 367-373, 1989.

[3] Baldi, Pierre and Kurt Hornik,"Neural Networks and Principal Component Analysis: Learning from Examples Without Local Minima", Neural Networks, Vol. 2,pp 53-58, 1989.

[4] Moody, John and Darken, Christen, " Learning with Localized Receptive Fields", Proc. of the 1988 Connectionist Models Summer School,CMU.

[5] Poggio Tomaso and Fedrico Giorsi,"Networks for Approximation and Learning", Proc. of IEEE, vol. 78, no. 9, September 1990, pp 1481- 1496.

[6] Saha, Avijit , D. S. Tang and Chuan-Lin Wu,."Dimension Reduction Using Networks of Linear Superposition of Gaussian Units",MCC Technical Report,, Sept. 1990.

[7] Moody, John and Darken, Christen, " Learning with Localized Receptive Fields", Proc. of the 1988 Connectionist Models Summer School, CMU.
